# Large Scale Nonparametric Bayesian Inference: Data Parallelisation in the Indian Buffet Process

**Finale Doshi-Velez**[*]
University of Cambridge
Cambridge, CB21PZ, UK
finale@alum.mit.edu

**David Knowles**[*]
University of Cambridge
Cambridge, CB21PZ, UK
dak33@cam.ac.uk

**Shakir Mohamed**[*]
University of Cambridge
Cambridge, CB21PZ, UK
sm694@cam.ac.uk

**Zoubin Ghahramani**
University of Cambridge
Cambridge, CB21PZ, UK
zoubin@eng.cam.ac.uk

## Abstract

Nonparametric Bayesian models provide a framework for flexible probabilistic modelling of complex datasets. Unfortunately, the high-dimensional averages required for Bayesian methods can be slow, especially with the unbounded representations used by nonparametric models. We address the challenge of scaling Bayesian inference to the increasingly large datasets found in real-world applications. We focus on parallelisation of inference in the Indian Buffet Process (IBP), which allows data points to have an unbounded number of sparse latent features. Our novel MCMC sampler divides a large data set between multiple processors and uses message passing to compute the global likelihoods and posteriors. This algorithm, the first parallel inference scheme for IBP-based models, scales to datasets orders of magnitude larger than have previously been possible.

## 1 Introduction

From information retrieval to recommender systems, from bioinformatics to financial market analysis, the amount of data available to researchers has exploded in recent years. While large, these datasets are often still sparse: For example, a biologist may have expression levels from thousands of genes from only a few people. A ratings database may contain millions of users and thousands of movies, but each user may have only rated a few movies. In such settings, Bayesian methods provide a robust approach to drawing inferences and making predictions from sparse information. At the heart of Bayesian methods is the idea that all unknown quantities should be averaged over when making predictions. Computing these high-dimensional average is thus a key challenge in scaling Bayesian inference to large datasets, especially for nonparametric models.

Advances in multicore and distributed computing provide one answer to this challenge: if each processor can consider only a small part of the data, then inference in these large datasets might become more tractable. However, such *data parallelisation* of inference is nontrivial—while simple models might only require pooling a small number of sufficient statistics [1], inference in more complex models might require the frequent communication of complex, high-dimensional probability distributions between processors. Building on work on approximate asynchronous multicore inference for topic models [2], we develop a message passing framework for data-parallel Bayesian inference applicable to a variety of models, including matrix factorization and the Indian Buffet Process (IBP).

---

[*] Authors contributed equally.

Nonparametric models are attractive for large datasets because they automatically adapt to the complexity of the data, relieving the researcher from the need to specify aspects of the model such as the number of latent factors. Much recent work in nonparametric Bayesian modelling has focused on the Chinese restaurant process (CRP), which is a discrete distribution that can be used to assign data points to an unbounded number of clusters. However, many real-world datasets have observations that may belong to multiple clusters—for example, a gene may have multiple functions; an image may contain multiple objects. The IBP [3] is a distribution over infinite sparse binary matrices that allows data points to be represented by an unbounded number of sparse latent features or factors. While the parallelisation method we present in this paper is applicable to a broad set of models, we focus on inference for the IBP because of its unique challenges and potential.

Many serial procedures have been developed for inference in the IBP, including variants of Gibbs sampling [3, 4], which may be augmented with Metropolis split-merge proposals [5], slice sampling [6], particle filtering [7], and variational inference [8]. With the exception of the accelerated Gibbs sampler of [4], these methods have been applied to datasets with less than 1,000 observations.

To achieve efficient paralellisation, we exploit an idea recently introduced in [4], which maintains a distribution over parameters while sampling. Coupled with a message passing scheme over processors, this idea enables computations for inference to be distributed over many processors with few losses in accuracy. We demonstrate our approach on a problem with 100,000 observations. The largest application of IBP inference to date, our work opens the use of the IBP and similar models to a variety of data-intensive applications.

## 2 Latent Feature Model

The IBP can be used to define models in which each observation is associated with a set of latent factors or features. A binary feature-assignment matrix $Z$ represents which observations possess which hidden features, where $Z_{nk} = 1$ if observation $n$ has feature $k$ and $Z_{nk} = 0$ otherwise. For example, the observations might be images and the hidden features could be possible objects in those images. Importantly, the IBP allows the set of such possible hidden features to be unbounded.

To generate a sample from the IBP, we first imagine that the rows of $Z$ (the observations) are customers and the columns of $Z$ (the features) are dishes in an infinite buffet. The first customer takes the first Poisson($\alpha$) dishes. The following customers try previously sampled dishes with probability $m_k/n$, where $m_k$ is the number of people who tried dish $k$ before customer $n$. Each customer also takes Poisson($\alpha/n$) new dishes. The value $Z_{nk}$ records if customer $n$ tried dish $k$. This generative process allows an unbounded set of features but guarantees that a finite dataset will contain a finite number of features with probability one. The process is also exchangeable in that the order in which customers visit the buffet has no impact on the distribution of $Z$. Finally, if the effect of possessing a feature is independent of the feature index, the model is also exchangeable in the columns of $Z$.

We associate with the feature assignment matrix $Z$, a feature matrix $A$ with rows that parameterise the effect that possessing each feature has on the data. Given these matrices, we write the probability of the data as $P(X|Z, A)$. Our work requires that $P(A|X, Z)$ can be computed or approximated efficiently by an exponential family distribution. Specifically, we apply our techniques to both a fully-conjugate linear-Gaussian model and non-conjugate Bernoulli model.

**Linear Gaussian Model.** We model an $N \times D$ real-valued data matrix $X$ as a product:

$$X = ZA + \epsilon, \tag{1}$$

where $Z$ is the binary feature-assignment matrix and $A$ is a $K$ by $D$ real-valued matrix with an independent Gaussian prior $N(0, \sigma_a^2)$ on each element (see cartoon in Figure 1(a)). Each element of the $N$ by $D$ noise matrix $\epsilon$ is independent with a $N(0, \sigma_n^2)$ distribution. Given $Z$ and $X$, the posterior on the features $A$ is Gaussian, given by mean and covariance

$$\mu^A = \left( Z^T Z + \frac{\sigma_x^2}{\sigma_a^2} I \right)^{-1} Z^T X \qquad \Sigma^A = \sigma_x^2 \left( Z^T Z + \frac{\sigma_x^2}{\sigma_a^2} I \right)^{-1} \tag{2}$$

**Bernoulli Model.** We use a leaky, noisy-or likelihood for each element of an $N \times D$ matrix $X$:

$$P(X_{nd} = 1|Z, A) = 1 - \epsilon \, \lambda^{\sum_k Z_{nk} A_{kd}}. \tag{3}$$

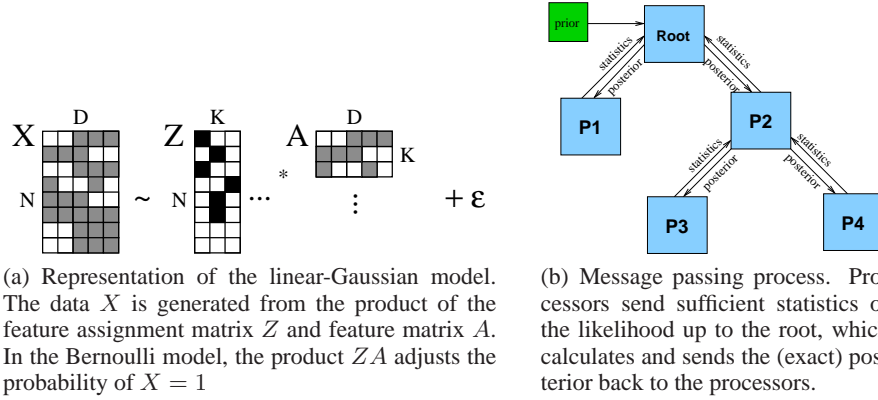

(a) Representation of the linear-Gaussian model. The data $X$ is generated from the product of the feature assignment matrix $Z$ and feature matrix $A$. In the Bernoulli model, the product $ZA$ adjusts the probability of $X = 1$

(b) Message passing process. Processors send sufficient statistics of the likelihood up to the root, which calculates and sends the (exact) posterior back to the processors.

Figure 1: Diagrammatic representation of the model structure and the message passing process.

Each element of the $A$ matrix is binary with independent Bernoulli($p_A$) priors. The parameters $\epsilon$ and $\lambda$ determine how "leaky" and how "noisy" the or-function is, respectively. Typical hyperparameter values are $\epsilon = 0.95$ and $\lambda = 0.2$. The posterior $P(A|X, Z)$ cannot be computed in closed form; however, a mean-field variational posterior in which we approximate $P(A|X, Z)$ as product of independent Bernoulli variables $\prod_{k,d}^{K,D} q_{kd}(a_{kd})$ can be readily derived.

## 3 Parallel Inference

We describe both synchronous and asynchronous procedures for approximate, parallel inference in the IBP that combines MCMC with message passing. We first partition the data among the processors, using $X^p$ to denote the subset of observations $X$ assigned to processor $p$. We use $Z^p$ to denote the latent features associated with the data on processor $p$. In [4], the distribution $P(A|X_{-n}, Z_{-n})$ was used to derive an accelerated sampler for sampling $Z_n$, where $n$ indexes the $n^{th}$ observation and $-n$ is the set of all observations except $n$. In our parallel inference approach, each processor $p$ maintains a distribution $P^p(A|X_{-n}, Z_{-n})$, a local approximation to $P(A|X_{-n}, Z_{-n})$. The distributions $P^p$ are updated via message passing between the processors.

The inference alternates between three steps:

- *Message passing:* processors communicate to compute the exact $P(A|X, Z)$.
- *Gibbs sampling:* processors sample a new set of $Z^p$'s in parallel.
- *Hyperparameter sampling:* a root processor resamples global hyperparameters

The sampler is approximate because during Gibbs sampling, all processors resample elements of $Z$ at the same time; their posteriors $P^p(A|X, Z)$ are no longer the true $P(A|X, Z)$.

**Message Passing** We use Bayes rule to factorise the posterior over features $P(A|Z, X)$:

$$P(A|Z, X) \propto P(A) \prod_p P(X^p|Z^p, A) \qquad (4)$$

If the prior $P(A)$ and the likelihoods $P(X^p|Z^p, A)$ are conjugate exponential family models, then the sufficient statistics of $P(A|Z, X)$ are the sum of the sufficient statistics of each term on the right side of equation (4). For example, the sufficient statistics in the linear-Gaussian model are means and covariances; in the Bernoulli model, they are counts of how often each element $A_{kd}$ equals one. The linear-Gaussian messages have size $O(K^2 + KD)$, and the Bernoulli messages $O(KD)$, where $K$ is the number of features. For nonparametric models such as the IBP, the number of features $K$ grows as $O(\log N)$. This slow growth means that messages remain small, even for large datasets.

The most straightforward way to compute the full posterior is to arrange processors in a tree architecture, as belief propagation is then exact. The message $s$ from processor $p$ to processor $q$ is:

$$s_{p \to q} = l^p + \sum_{r \in N(p) \backslash q} s_{r \to p}$$

where $N(p)\backslash q$ are the processors attached to $p$ besides $q$ and $l^p$ are the sufficient statistics from processor $p$. A dummy neighbour containing the statistics of the prior is connected to (an arbitrarily designated) root processor. Also passed are the feature counts $m_k^p = \sum_{n \in X^p} Z_{nk}^p$, the popularity of feature $k$ within processor $p$. (See figure 1(b) for a cartoon.)

**Gibbs Sampling**    In general, $Z_{nk}$ can be Gibbs-sampled using Bayes rule

$$P(Z_{nk}|Z_{-nk}, X) \propto P(Z_{nk}|Z_{-nk})P(X|Z).$$

The probability $P(Z_{nk}|Z_{-nk})$ depends on the size of the dataset $N$ and the number of observations $m_k$ using feature $k$. At the beginning of the Gibbs sampling stage, each processor has the correct values of $m_k$. We compute $m_k^{-p} = m_k - m_k^p$, and, as the processor's internal feature counts $m_k^p$ are updated, approximate $m_k \approx m_k^{-p} + m_k^p$. This approximation assumes $m_k^{-p}$ stays fixed during the current stage (good for popular features).

The *collapsed* likelihood $P(X|Z)$ integrating out the feature values $A$ is given by:

$$P(X|Z) \propto \int_A P(X_n|Z_n, A)P(A|Z_{-n}, X_{-n})dA,$$

where the partial posterior $P(A|Z_{-n}, X_{-n}) \propto \frac{P(A|Z,X)}{P(X_n|Z_n,A)}$. In conjugate models, $P(A|Z_{-n}, X_{-n})$ can be efficiently computed by subtracting observation $n$'s contribution to the sufficient statistics.[1] For non-conjugate models, we can use an exponential family distribution $Q(A)$ to approximate $P(A|X, Z)$ during message passing. A draw $A \sim Q^{-p}(A)$ is then used to initialise an *uncollapsed* Gibbs sampler. The outputted samples of $A$ are used to compute sufficient statistics for the likelihood $P(X|Z)$. In both cases, new features are added as described in [3].

**Hyperparameter Resampling**    The IBP concentration parameter $\alpha$ and hyperparameters of the likelihood can also be sampled during inference. Resampling $\alpha$ depends only on the total number of active features; thus it can easily be resampled at the root and propagated to the other processors. In the linear-Gaussian model, the posteriors on the noise and feature variances (starting from gamma priors) depend on various squared-errors, which can also be computed in a distributed fashion.

For more general, non-conjugate models, resampling the hyperparameters requires two steps. In the first step, a hyperparameter value is proposed by the root and propagated to the processors. The processors each compute the likelihood of the current and proposed hyperparameter values and propagate this value back to root. The root evaluates a Metropolis step for the hyperparameters and propagates the decision back to the leaves. The two-step approach introduces a latency in the resampling but does not require any additional message passing rounds.

**Asynchronous Operation**    So far we have discussed message passing, Gibbs sampling, and hyperparameter resampling as if they occur in separate phases. In practice, these phases may occur asynchronously: between its Gibbs sweeps, each processor updates its feature posterior based on the most current messages it has received and sends likelihood messages to its parent. Likewise, the root continuously resamples hyperparameters and propagates the values down through the tree. While another layer of approximation, this asynchronous form of message passing allows faster processors to share information and perform more inference on their data instead of waiting for slower processors.

**Implementation Note**    When performing parallel inference in the IBP, a few factors need to be considered with care. Other parallel inference for nonparametric models, such as the HDP [2], simply matched features by their index, that is, assumed that the $i^{th}$ feature on processor $p$ was also the $i^{th}$ feature on processor $q$. In the IBP, we find that this indiscriminate feature merging is often disastrous when adding or deleting features: if none of the observations in a particular processor are using a feature, we cannot simply delete that column of Z and shift the other features over—doing so destroys the alignment of features across processors.

# 4 Comparison to Exact Metropolis

Because all $Z^p$'s are sampled at once, the posteriors $P^p(A|X, Z)$ used by each processor in section 3 are no longer exact. Below we show how Metropolis–Hastings (MH) steps can make the parallel sampler exact, but introduce significant computational overheads both in computing the transition probabilities and in the message passing. We argue that trying to do exact inference is a poor use of computational resources (especially as any finite chain will not be exact); empirically, the approximate sampler behaves similarly to the MH sampler while finding higher likelihood regions in the data.

**Exact Parallel Metropolis Sampler.** Ideally, we would simply add an MH accept/reject step after each stage of the approximate inference to make the sampler exact. Unfortunately, the approximate sampler makes several non-independent random choices in each stage of the inference, making the reverse proposal inconvenient to compute. We circumvent this issue by fixing the random seed, making the initial stage of the approximate sampler a deterministic function, and then add independent random noise to create a proposal distribution. This approach makes both the forward and reverse transition probabilities simple to compute.

Formally, let $\hat{Z}^p$ be the matrix output after a set of Gibbs sweeps on $Z^p$. We use all the $\hat{Z}^p$'s to propose a new $Z'$ matrix. The acceptance probability of the proposal is

$$\min(1, \frac{P(X|Z')P(Z')Q(Z' \to Z)}{P(X|Z)P(Z)Q(Z \to Z')}),$$ (5)

where the likelihood terms $P(X|Z)$ and $P(Z)$ are readily computed in a distributed fashion. For the transition distribution $Q$, we note that if we set the random seed $r$, then the matrix $\hat{Z}^p$ from the Gibbs sweeps in the processor is some deterministic function of the input matrix $Z^p$. The proposal $Z^{p\prime}$ is a (stochastic) noisy representation of $\hat{Z}^p$ in which for example

$$P(Z_{nk}^{p\prime} = 1) = .99 \quad \texttt{if} \quad \hat{Z}_{nk}^p = 1, \qquad P(Z_{nk}^{p\prime} = 1) = .01 \quad \texttt{if} \quad \hat{Z}_{nk}^p = 0$$ (6)

where $K$ should be at least the number of features in $\hat{Z}^p$. We set $Z_{nk}^{p\prime} = 0$ for $k > K$. (See cartoon in figure 2.)

To compute the backward probability, we take $Z^{p\prime}$ and apply the same number of Gibbs sampling sweeps with the same random seed $r$. The resulting $\hat{Z}^{p\prime}$ is a deterministic function of $Z^{p'}$. The backward probability $Q(Z^{p'} \to Z^p)$ which is the probability of going from $Z^{p\prime}$ to $Z^p$ using 6. While the transition probabilities can be computed in a distributed, asynchronous fashion, all of the processors must synchronise when deciding whether to accept the proposal.

**Experimental Comparison** To compare the exact Metropolis and approximate inference techniques, we ran each inference type on 1000 block images of [3] on 5 simulated processors. Each test was repeated 25 times. For each of the 25 tests, we create a held out dataset by setting elements of the last 100 images as missing values. For the first 50 test images, we set all even numbered dimensions as the missing elements, and every odd numbered dimension as the missing values for the last 50 images. Each sampler was run for 10,000 iterations with 5 Gibbs sweeps per iteration; statistics were collected from the second half of the chain. To keep the probability of an acceptance reasonable, we allowed each processor to change only small parts of its $Z^p$: the feature assignments $Z_n$ for 1, 5, or 10 data points each during each sweep.

In table 1, we see that the approximate sampler runs about five times faster than the exact samplers while achieving comparable (or better) predictive likelihoods and reconstruction errors on held-out data. Both the acceptance rates and the predictive likelihoods fall as the exact sampler tries to take larger steps, suggesting that the difference between the approximate and exact sampler's performance on predictive likelihood is due to poor mixing by the exact sampler. Figure 4 shows empirical CDFs for the number of features $k$, IBP concentration parameter $\alpha$, the noise variance $\sigma_n^2$, and the feature variance $\sigma_a^2$. The approximate sampler (black) produces similar CDFs to the various exact Metropolis samplers (gray) for the variances; the concentration parameter is smaller, but the feature counts are similar to the single-processor case.

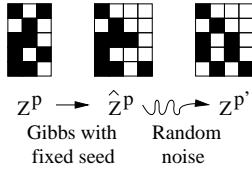

Figure 2: Cartoon of MH proposal

| Method | Time (s) | Test $L_2$ Error | Test Log Likelihood | MH Accept Proportion |
|---|---|---|---|---|
| MH, n = 1 | 717 | 0.0468 | 0.1098 | 0.1106 |
| MH, n = 5 | 1075 | 0.0488 | 0.0893 | 0.0121 |
| MH, n = 10 | 1486 | 0.0555 | 0.0196 | 0.0062 |
| Approximate | 179 | 0.0487 | 0.1292 | - |

Table 1: Evaluation of exact and approximate methods.

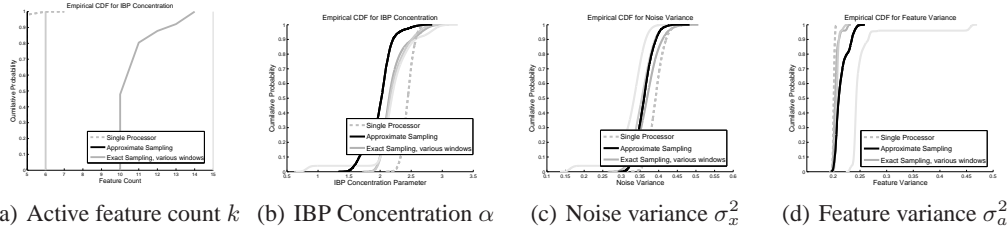

(a) Active feature count $k$    (b) IBP Concentration $\alpha$    (c) Noise variance $\sigma_x^2$    (d) Feature variance $\sigma_a^2$

Figure 3: Empirical CDFs: The solid black line is the approximate sampler; the three solid gray lines are the MH samplers with $n$ equal to 1, 5, and 10 (lighter shades indicate larger $n$. The approximate sampler and the MH samplers for smaller $n$ have similar CDFs; the $n = 10$ MH sampler's differing CDF indicates it did not mix in 7500 iterations (reasonable since its acceptance rate was 0.0062).

## 5 Analysis of Mixing Properties

We ran a series of experiments on 10,000 36-dimensional block images of [3] to study the effects of various sampler configurations on running time, performance, and mixing time properties of the sampler. 5000 elements of the data matrix were held-out as test data. Figure 4 shows test log-likelihoods using 1, 7, 31 and 127 parallel processors simulated in software, using 1000 outer iterations with 5 Gibbs inner iterations each. The parallel samplers have similar test likelihoods as the serial algorithm with significant savings in running time. The characteristic shape of the test likelihood, similar across all testing regimes, indicates how the features are learned. Initially, a large number of features are added, which provides improvements in the test likelihood. A refinement phase, in which excess features are pruned, provides further improvements.

Figure 4 shows hairiness-index plots for each of the test cases after thinning and burn-in. The hairiness index, based on the method of CUSUM for monitoring MCMC convergence [9, 10], monitors how often the derivatives of sampler statistics—in our case, the number of features, the test likelihood, and $\alpha$—change in sign; infrequent changes in sign indicate that the sampler may not be mixed. The outer bounds on the plots are the 95% confidence bounds. The index stays within the bounds suggesting that the chains are mixing.

Finally, we considered the trade-off between mixing and running time as the number of outer iterations and inner Gibbs iterations are varied. Each combination of inner and outer iterations was set so that the total number of Gibbs sweeps through the data was 5000. Mixing efficiency was

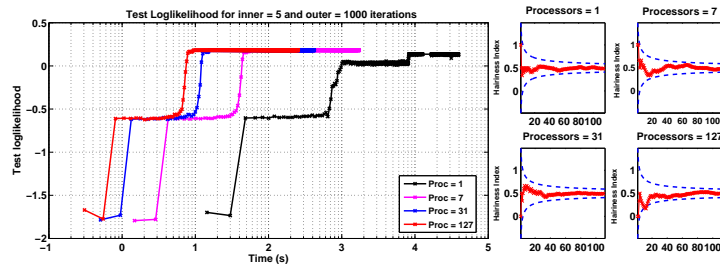

Figure 4: Change in likelihood for various numbers of processors over the simulation time. The corresponding hairiness index plots are shown on the left.

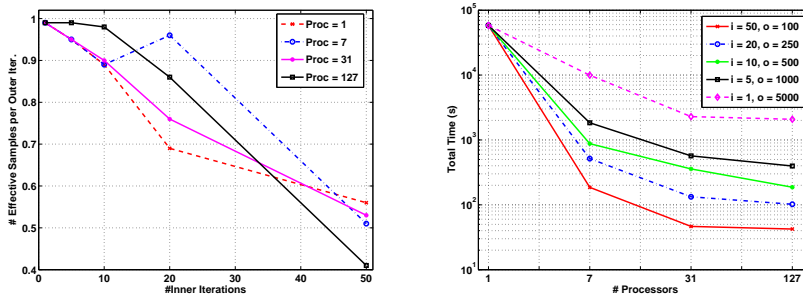

Figure 5: Effects of changing the number of inner iterations on: (a) The effective sample size (b) Total running time (Gibbs and Message passing).

Table 2: Test log-likelihoods on real-world datasets for the serial, synchronous and asynchronous inference types.

| Dataset | N | D | Description | Serial $p = 1$ | Synch $p = 16$ | Async $p = 16$ |
|---------|---|---|-------------|----------------|----------------|----------------|
| AR Faces [11] | 2600 | 1598 | faces with lighting, accessories (real-valued) | -4.74 | -4.77 | -4.84 |
| Piano [12] | 57931 | 161 | STDFT of a piano recording (real-valued) | -1.435 | -1.182 | -1.228 |
| Flickr [13] | 100000 | 1000 | indicators of image tags (binary-valued) | — | -0.0584 | |

measured via the effective number of samples per sample [10], which evaluates what fraction of the samples are independent (ideally, we would want all samples to be independent, but MCMC produces dependent chains). Running time for Gibbs sampling was taken to be the time required by the slowest processor (since all processors must synchronize before message passing); the total time reflected the Gibbs time and the message-passing time. As seen in figure 5, completing fewer inner Gibbs iterations per outer iteration results in faster mixing, which is sensible as the processors are communicating about their data more often. However, having fewer inner iterations requires more frequent message passing; as the number of processors becomes large, the cost of message passing becomes a limiting factor.[2]

## 6   Real-world Experiments

We tested our parallel scheme on three real world datasets on a 16 node cluster using the Matlab Distributed Computing Engine, using 3 inner Gibbs iterations per outer iteration. The first dataset was a set of 2,600 frontal face images with 1,598 dimensions [11]. While not extremely large, the high-dimensionality of the dataset makes it challenging for other inference approaches. The piano dataset [12] consisted of 57,931 samples from a 161-dimensional short-time discrete Fourier transform of a piano piece. Finally, the binary-valued Flickr dataset [13] indicated whether each of 1000 popular keywords occurred in the tags of 100,000 images from Flickr. Performance was measured using test likelihoods and running time. Test likelihoods look only at held-out data and thus they allow us to 'honestly' evaluate the model's fit. Table 2 summarises the data and shows that all approaches had similar test-likelihood performance.

In the faces and music datasets, the Gibbs time per iteration improved almost linearly as the number of processors increased (figure 6). For example, we observed a 14x-speedup for $p = 16$ in the music dataset. Meanwhile, the message passing time remained small even with 16 processors—7% of the Gibbs time for the faces data and 0.1% of the Gibbs time for the music data. However, waiting for synchronisation became a significant factor in the synchronous sampler. Figure 6(c) compares the times for running inference serially, synchronously and asynchronously with 16 processors. The

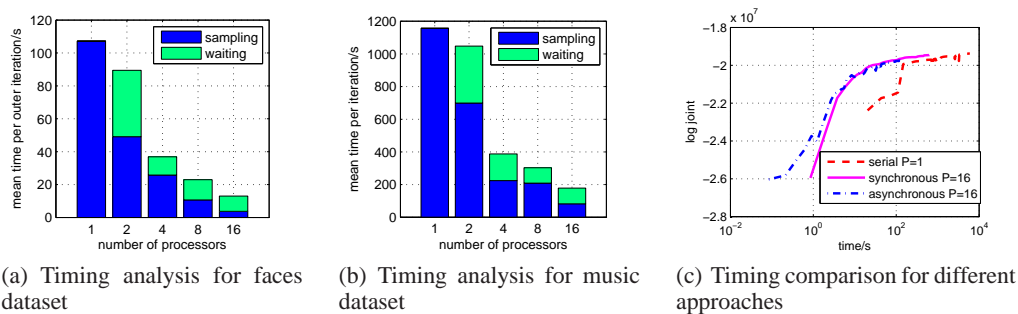

(a) Timing analysis for faces dataset

(b) Timing analysis for music dataset

(c) Timing comparison for different approaches

Figure 6: Bar charts comparing sampling time and waiting times for synchronous parallel inference.

asynchronous inference is 1.64 times faster than the synchronous case, reducing the computational time from 11.8s per iteration to 7.2s.

# 7 Discussion and Conclusion

As datasets grow, parallelisation is an increasingly attractive and important feature for doing inference. Not only does it allow multiple processors/multicore technologies to be leveraged for large-scale analyses, but it also reduces the amount of data and associated structures that each processor needs to keep in memory. Existing work has focused both on general techniques to efficiently split variables across processors in undirected graphical models [14] and factor graphs [15] and specific models such as LDA [16, 17]. Our work falls in between: we leverage properties of a specific kind of parallelisation—data parallelisation—for a fairly broad class of models.

Specifically, we describe a parallel inference procedure that allows nonparametric Bayesian models based on the Indian Buffet Process to be applied to large datasets. The IBP poses specific challenges to data parallelisation in that the dimensionality of the representation changes during inference and may be unbounded. Our contribution is an algorithm for data-parallelisation that leverages a compact representation of the feature posterior that approximately decorrelates the data stored on each processor, thus limiting the communication bandwidth between processors. While we focused on the IBP, the ideas presented here are applicable to a more general problems in unsupervised learning including bilinear models such as PCA, NMF, and ICA.

Our sampler is approximate, and we show that in conjugate models, it behaves similarly to an exact sampler—but with much less computational overhead. However, as seen in the Bernoulli case, variational message passing for non-conjugate data doesn't always produce good results if the approximating distribution is a poor match for the true feature posterior. Determining when variational message passing is successful is an interesting question for future work. Other interesting directions include approaches for dynamically optimising the network topology (for example, slower processors could be moved lower in the tree). Finally, we note that a middle ground between synchronous and asynchronous operations as we presented them might be a system that gives each processor a certain amount of time, instead of a certain number of iterations, to do Gibbs sweeps. Further study along these avenues should lead to even more efficient data-parallel Bayesian inference techniques.

## Footnotes

[1]In the IBP, only the linear-Gaussian model exhibits this conjugate structure. However, many other matrix factorization models (such as PCA) often have this conjugate form.

[2]We believe part of the timing results may be an artifact, as the simulation overestimates the message passing time. In the actual parallel system (section 6), the cost of message passing was negligible.

# References

[1] C. Chu, S. Kim, Y. Lin, Y. Yu, G. Bradski, A. Ng, and K. Olukotun, "Map-reduce for machine learning on multicore," in *Advances in Neural Information Processing Systems*, p. 281, MIT Press, 2007.

[2] A. Asuncion, P. Smyth, and M. Welling, "Asynchronous distributed learning of topic models," in *Advances in Neural Information Processing Systems 21*, 2008.

[3] T. Griffiths and Z. Ghahramani, "Infinite latent feature models and the Indian buffet process," in *Advances in Neural Information Processing Systems*, vol. 16, NIPS, 2006.

[4] F. Doshi-Velez and Z. Ghahramani, "Accelerated inference for the Indian buffet process," in *International Conference on Machine Learning*, 2009.

[5] E. Meeds, Z. Ghahramani, R. Neal, and S. Roweis, "Modeling dyadic data with binary latent factors," in *Advances in Neural Information Processing Systems*, vol. 19, pp. 977–984, 2007.

[6] Y. W. Teh, D. Görür, and Z. Ghahramani, "Stick-breaking construction for the Indian buffet process," in *Proceedings of the Intl. Conf. on Artificial Intelligence and Statistics*, vol. 11, pp. 556–563, 2007.

[7] F. Wood and T. L. Griffiths, "Particle filtering for nonparametric Bayesian matrix factorization," in *Advances in Neural Information Processing Systems*, vol. 19, pp. 1513–1520, 2007.

[8] F. Doshi-Velez, K. T. Miller, J. Van Gael, and Y. W. Teh, "Variational inference for the Indian buffet process," in *Proceedings of the Intl. Conf. on Artificial Intelligence and Statistics*, vol. 12, pp. 137–144, 2009.

[9] S. P. Brooks and G. O. Roberts, "Convergence assessment techniques for Markov Chain Monte Carlo," *Statistics and Computing*, vol. 8, pp. 319–335, 1998.

[10] C. R. Robert and G. Casella, *Monte Carlo Statistical Methods*. Springer, second ed., 2004.

[11] A. M. Mart'inez and A. C. Kak, "PCA versus LDA," *IEEE Trans. Pattern Anal. Mach. Intelligence*, vol. 23, pp. 228–233, 2001.

[12] G. E. Poliner and D. P. W. Ellis, "A discriminative model for polyphonic piano transcription," *EURASIP J. Appl. Signal Process.*, vol. 2007, no. 1, pp. 154–154, 2007.

[13] T. Kollar and N. Roy, "Utilizing object-object and object-scene context when planning to find things.," in *International Conference on Robotics and Automation*, 2009.

[14] C. G. Joseph Gonzalez, Yucheng Low, "Residual splash for optimally parallelizing belief propagation," in *Proceedings of the Twelfth International Conference on Artificial Intelligence and Statistics* (D. van Dyk and M. Welling, eds.), vol. 5, pp. 177–184, JMLR, 2009.

[15] D. Stern, R. Herbrich, and T. Graepel, "Matchbox: Large scale online Bayesian recommendations," in *18th International World Wide Web Conference (WWW2009)*, April 2009.

[16] R. Nallapati, W. Cohen, and J. Lafferty, "Parallelized variational EM for Latent Dirichlet Allocation: An experimental evaluation of speed and scalability," in *ICDMW '07: Proceedings of the Seventh IEEE International Conference on Data Mining Workshops*, (Washington, DC, USA), pp. 349–354, IEEE Computer Society, 2007.

[17] D. Newman, A. Asuncion, P. Smyth, and M. Welling, "Distributed inference for Latent Dirichlet Allocation," in *Advances in Neural Information Processing Systems 20* (J. Platt, D. Koller, Y. Singer, and S. Roweis, eds.), pp. 1081–1088, Cambridge, MA: MIT Press, 2008.

